# Cell Assemblies in Large Sparse Inhibitory Networks of Biologically Realistic Spiking Neurons

**Adam Ponzi**
OIST, Uruma, Okinawa, Japan.
adamp@oist.jp

**Jeff Wickens**
OIST, Uruma, Okinawa, Japan.
wickens@oist.jp

## Abstract

Cell assemblies exhibiting episodes of recurrent coherent activity have been observed in several brain regions including the striatum[1] and hippocampus CA3[2]. Here we address the question of how coherent dynamically switching assemblies appear in large networks of biologically realistic spiking neurons interacting deterministically. We show by numerical simulations of large asymmetric inhibitory networks with fixed external excitatory drive that if the network has intermediate to sparse connectivity, the individual cells are in the vicinity of a bifurcation between a quiescent and firing state and the network inhibition varies slowly on the spiking timescale, then cells form assemblies whose members show strong positive correlation, while members of different assemblies show strong negative correlation. We show that cells and assemblies switch between firing and quiescent states with time durations consistent with a power-law. Our results are in good qualitative agreement with the experimental studies. The deterministic dynamical behaviour is related to winner-less competition[3], shown in small closed loop inhibitory networks with heteroclinic cycles connecting saddle-points.

## 1 Introduction

Cell assemblies exhibiting episodes of recurrent coherent activity have been observed in several brain regions including the striatum[1] and hippocampus CA3[2], but how such correlated activity emerges in neural microcircuits is not well understood. Here we address the question of how coherent assemblies can emerge in large inhibitory neural networks and what this implies for the structure and function of one such network, the striatum.

Carrillo-Reid et al.[1] performed calcium imaging of striatal neuronal populations and revealed sporadic and asynchronous activity. They found that burst firing neurons were widespread within the field of observation and that sets of neurons exhibited episodes of recurrent and synchronized bursting. Furthermore dimensionality reduction of network dynamics revealed functional states defined by cell assemblies that alternated their activity and displayed spatiotemporal pattern generation. Recurrent synchronous activity traveled from one cell assembly to the other often returning to the original assembly; suggesting a robust structure. Assemblies were visited non-randomly in sequence and not all state transitions were allowed. Moreover the authors showed that while each cell assembly comprised different cells, a small set of neurons was shared by different assemblies. Although the striatum is an inhibitory network composed of GABAergic projection neurons, similar types of cell assemblies have also been observed in excitatory networks such as the hippocampus. In a related and similar study Sasaki et al.[2] analysed spontaneous CA3 network activity in hippocampal slice cultures using principal component analysis. They found discrete heterogeneous network states defined by active cell ensembles which were stable against external perturbations through synaptic activity. Networks tended to remain in a single state for tens of seconds and then suddenly jump to a new state. Interestingly the authors tried to model the temporal profile of state transitions by a

hidden Markov model, but found that the transitions could not be simulated in this way. The authors suggested that state dynamics is non-random and governed by local attractor-like dynamics.

We here address the important question of how such assemblies can appear deterministically in biologically realistic cell networks. We focus our modeling on the inhibitory network of the striatum, however similar models can be proposed for networks such as CA3 if the cell assembly activity is controlled by the inhibitory CA3 interneurons. Network synchronization dynamics[4, 5] of random sparse inhibitory networks of CA3 interneurons has been addressed by Wang and Buzsaki[5]. They determined specific conditions for population synchronization including that the ratio between the synaptic decay time constant and the oscillation period be sufficiently large and that a critical minimal average number of synaptic contacts per cell, which was not sensitive to the network size, was required. Here we extend this work focusing on the formation of burst firing cell assemblies

The striatum is composed of GABAergic projection neurons with fairly sparse asymmetric inhibitory collaterals which seem quite randomly structured and that receive an excitatory cortical projection[6]. Each striatal medium spiny neuron (MSN) is inhibited by about 500 other MSNs in the vicinity via these inhibitory collaterals and similarly each MSN inhibits about 500 MSNs. However only about $10\% - 30\%$ of MSNs are actually excited by cortex at any particular time. This implies that each MSN is actively inhibited by about $50 - 150$ cortically excited cells in general. It is important to understand why the striatum has this particular structure, which is incompatible with its putative winner-take-all role. We show by numerical computer simulation that very general random networks of biologically realistic neurons coupled with inhibitory Rall-type synapses[7] and individually driven by excitatory input can show switching assembly dynamics. We commonly observed a switching bursting regime in networks with sparse to intermediate connectivity when the level of network inhibition approximately balanced the external excitation so that the individual cells were near a bifurcation point. In our simulations, cells and assemblies slowly and spontaneously switch between a depolarized firing state and a more hyperpolarized quiescent state. The proportion of switching cells varies with the network connectivity, peaking at low connection probability for fixed total inhibition. The sorted cross correlation matrix of the firing rates time series for switching cells shows a fascinating multiscale clustered structure of cell assemblies similar to observations in[1, 2].

The origin of the deterministic switching dynamics in our model is related to the principle of *winnerless competition* (WLC) which has previously been observed by Rabinovich and coworkers[3] in small inhibitory networks with closed loops based on heteroclinic cycles connecting saddle points. Rabinovich and coworkers[3] demonstrate that such networks can generate stimulus specific patterns by switching among small and dynamically changing neural ensembles with application to insect olfactory coding[8, 9], sequential decision making[10] and central pattern generation[11]. Networks produce this switching mode of dynamical activity when lateral inhibitory connections are strongly non-symmetric. WLC can represent information dynamically and is reproducible, robust against intrinsic noise and sensitive to changes in the sensory input. A closely related dynamical phenomenon is referred to as *chaotic itinerancy,*[12]. This is a state that switches between fully developed chaos and ordered behavior. The orbit remains in the vicinity of lower dimensional quasi-stable nearly periodic "attractor ruins" for some time before eventually exiting to a state of high dimensional chaos. This high-dimensional state is also quasi-stable, and after chaotic wandering the orbit is again attracted to one of the attractor ruins. Our study suggests attractor switching may be ubiquitous in biologically realistic large sparse random inhibitory networks.

## 2   Model

The network is composed of biologically realistic model neurons in the vicinity of a bifurcation from a stable fixed point to spiking limit cycle dynamical behaviour. To describe the cells we use the $I_{Na,p} + I_k$ model described in Izhikevich[13] although any model near such a bifurcation would be appropriate. The $I_{Na,p} + I_k$ cell model is two-dimensional and described by,

$$C\frac{dV_i}{dt} = I_i(t) - g_L(V_i - E_L) - g_{Na}m_\infty(V_i)(V_i - E_{Na}) - g_k n_i(V_i - E_k) \qquad (1)$$

$$\frac{dn_i}{dt} = (n_\infty - n_i)/\tau_n \qquad (2)$$

having leak current $I_L$, persistent $Na^+$ current $I_{Na,p}$ with instantaneous activation kinetic and a relatively slower persistent $K^+$ current $I_K$. $V_i(t)$ is the membrane potential of the $i - th$ cell, $C$

the membrane capacitance, $E_{L,Na,k}$ are the channel reversal potentials and $g_{L,Na,k}$ are the maximal conductances. $n_i(t)$ is $K^+$ channel activation variable of the $i-th$ cell. The steady state activation curves $m_\infty$ and $n_\infty$ are both described by, $x_\infty(V) = 1/(1+\exp\{(V^x_\infty - V)/k^x_\infty\})$ where $x$ denotes $m$ or $n$ and $V^x_\infty$ and $k^x_\infty$ are fixed parameters. $\tau_n$ is the fixed timescale of the $K^+$ activation variable. The term $I_i(t)$ is the input current to the $i-th$ cell.

The parameters are chosen so that the cell is the vicinity of a *saddle-node on invariant circle* bifurcation. As the current $I_i(t)$ in Eq.1 increases through the bifurcation point the stable node fixed point and the unstable saddle fixed point annihilate each other and a limit cycle having zero frequency is formed[13]. Increasing current further increases the frequency of the limit cycle. The input current $I_i(t)$ in Eq.1 is composed of both excitatory and inhibitory parts and given by,

$$I_i(t) = I_i^c + \sum_j -k_{syn,ij}g_j(t)(V_i(t) - V_{syn}).$$ (3)

The excitatory part is represented by $I_i^c$ and models the effect of the cortico-striatal synapses. It has a fixed magnitude for the duration of a simulation, but varying across cells. In the simulations reported here the $I_i^c$ are quenched random variables drawn uniformly randomly from the interval $[I_{bif}, I_{bif} + 1]$ where $I_{bif} = 4.51$ is the current at the saddle-node bifurcation point. These values of excitatory input current mean that all cells would be on limit cycles and firing with low rates if the network inhibition were not present. In fact the inhibitory network may cause some cells to become quiescent by reducing the total input current to below the bifurcation point. Since the inhibitory current part is provided by the GABAergic collaterals of the striatal network it is dynamically variable. These synapses are described by Rall-type synapses[7] in Eq.3 where the current into postsynaptic neuron $i$ is summed over all inhibitory presynaptic neurons $j$ and $V_{syn}$ and $k_{syn,ij}$ are channel parameters. $g_j(t)$ is the quantity of postsynaptically bound neurotransmitter given by,

$$\tau_g \frac{dg_j}{dt} = \Theta(V_j(t) - V_{th}) - g_j(t)$$ (4)

for the $j-th$ presynaptic cell. Here $V_{th}$ is a threshold, and $\Theta(x)$ is the Heaviside function. $g_j$ is essentially a low-pass filter of presynaptic firing. The timescale $\tau_g$ should be set relatively large so that the postsynaptic conductance follows the exponentially decaying time average of many preceding presynaptic high frequency spikes.

The network structure is described by the parameters $k_{syn,ij} = (k_{syn}/p)\epsilon_{ij}X_{ij}$ where $\epsilon_{ij}$ is another uniform quenched random variable on $[0.5, 1.5]$ independent in $i$ and $j$. $X_{ij} = 1$ if cells $i$ and $j$ are connected and zero otherwise. In the simulations reported here we use random networks where cells $i$ and $j$ are connected with probability $p$, and there are no self-connections, $X_{ii} = 0$. $k_{syn}$ is a parameter which is rescaled by the connection probability $p$ so that average total inhibition on each cell is constant independent of $p$. All simulations were carried out with fourth order Runge-Kutta.

## 3 Results

Figure 1(a) shows a time series segment of membrane potentials $V_i(t)$ for some randomly selected cells from an $N = 100$ cell network. The switching between firing and quiescent states can clearly be seen. Cells fire with different frequencies and become quiescent for variable periods before starting to fire again apparently randomly. However the model has no stochastic variables and therefore this switching is caused by deterministic chaos. As explained above the firing rate is determined by the proximity of the limit cycle to the saddle-node bifurcation and can therefore be arbitrarily low for this type of bifurcation. Since we have set the unit parameters so that all units are near the bifurcation point even weak network inhibition is able to cause the cells to become quiescent at times. The parameter settings are biologically realistic[13] and MSN cells are known to show irregular quiescent and firing states in vivo[14].

The complex bursting structure is easier to see from raster plots. A segment from a $N = 100$ cell time series is shown in Fig.1(b). This figure clearly shows attractor switching, or chaotic itinerancy[12], where a quasi-stable nearly-periodic state (an "attractor ruin") is visited from higher dimensional chaos. To make this plot the cells have been ordered by the k-means algorithm with five clusters (see below). The cells are coloured according to the cluster assigned to them by the algorithm. During the periodic window, most cells are silent however some cells fire continuously

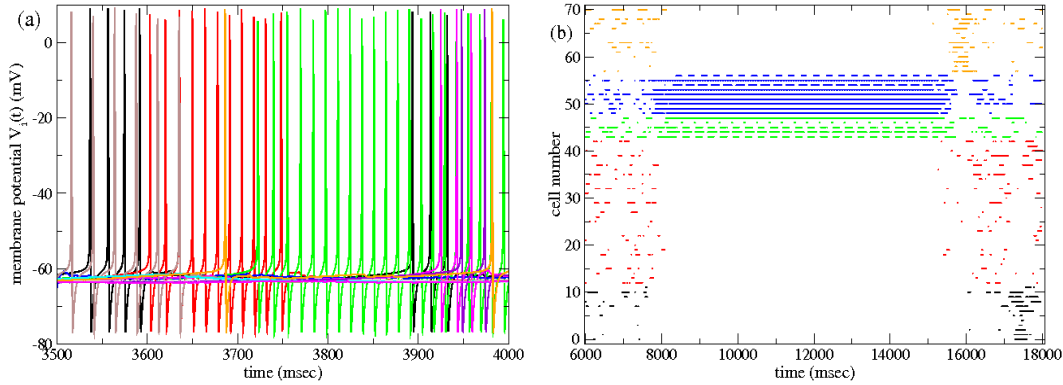

Figure 1: (a) Membrane potential $V_i(t)$ time series segment for a few cells from a $N = 100$ cell network simulation with 20 connections per cell. Each cell time series is a different colour. (b) Spike raster plot from an $N = 100$ cell network simulation with 20 connections per cell. Each line is a different cell and the 71 cells which fire at least one spike during the period shown are plotted. Cells are ordered by k-means with five clusters and coloured according to their assigned clusters.

at fixed frequency and some cells fire in periodic bursts. In fact the cells which fire in bursts have been separated into two clusters, as can be seen in Fig.1(b), the blue and green clusters. These two clusters fire periodic bursts in anti-phase. Cell assemblies can also be seen in the chaotic regions. The cells in the black cluster fire together in a burst around $t = 17500$ while the cells in the orange cluster fire a burst together around $t = 16000$. Fig.2(a) shows another example of a spike raster plot from a $N = 100$ cell network simulation where again the cells have been ordered by the k-means algorithm with five clusters. Now cell assemblies, blue, orange and red coloured, can clearly be seen which appear to switch in alternation. This switching is further interrupted from time to time by the green and black assemblies.

Due to the presence of attractor switching where cell assemblies can burst in antiphase we can expect the appearance of strongly positively and strongly negatively correlated cell pairs. Correlation matrices are constructed by dragging a moving window over a long spike time series and counting the spikes to construct the associated firing rate time series. The correlation matrix of the rate time series is then sorted by the k-means method[2], which is equivalent to PCA. Each cell is assigned to one of a fixed number of clusters and the cells indices are reordered accordingly. Fig.2(b) shows the cross-correlation matrix corresponding to the spike raster plot in Fig.2(a) with cells ordered the same way. Within an assembly cells are positively correlated, while cells in different assemblies often show negative correlation.

Larger networks with appropriate connectivites also show complex identity-temporal patterns. A patch-work of switching cell assembly clusters can be seen in the spike raster plot and corresponding cross-correlation matrix shown in Figs.2(c) and (d) respectively for a $N = 500$ cell system where the cells have been ordered by the k-means algorithm, now with 30 clusters. Any particular assembly can seem to be burst firing periodically for a spell before becoming quiescent for long spells. Other cell assemblies burst very occasionally for no apparent reason. Notice from the cross-correlation matrices in Fig.2(b) and (d) that although some cell assemblies are positively correlated with each other, they have different relationships to other cell assemblies, and therefore cannot be combined into a single larger assemblies.

Fig.2(c) reveals many cells switching between a firing state and quiescent state. What is the structure of this switching state? To investigate this we analyse inter spike interval (ISI) distributions. Shown in Fig.1(b) are three ISI distributions for three 500 cell network simulations in the sparse to intermediate regime with 30 connections per cell. The distributions are very broad and far from the exponential distribution one would expect from a Poisson process. They are consistent with a scale-free power law behaviour for three orders of magnitude, but exponentially cut off at large ISIs due to finite size effects. It is this distribution which produces the appearance of the complex identity-temporal patterns shown in the 500 cell time series figure in Fig.2(c) with the long ISIs interspersed with the bursts of short ISIs. Power-law distributions are characteristic of systems showing chaotic

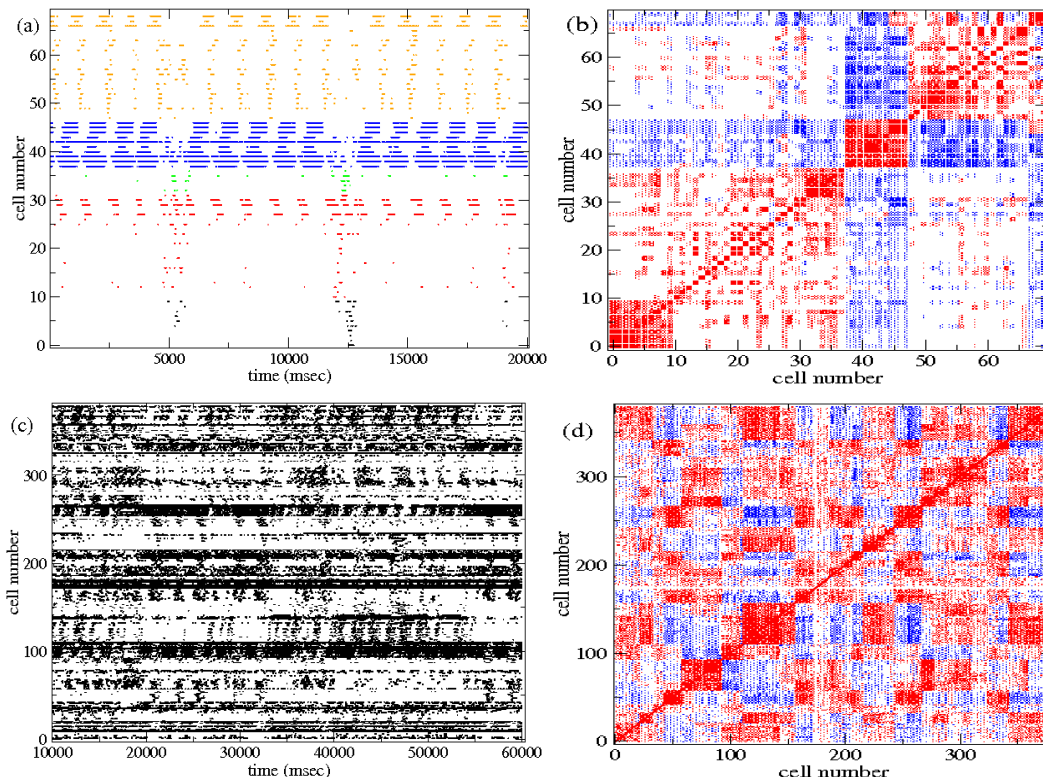

Figure 2: (a) Spike raster plot from all 69 cells in a 100 cell network with 20 connections per cell which fire at least one spike. The cells are ordered by k-means with five clusters and coloured according to their assigned cluster. (b) Cross-correlation matrix corresponding to (a). The cells are ordered by the k-means algorithm the same way as (a). Red colour means positive correlation, blue means negative correlation, colour intensity matches strength. White is weak or no correlation. (c) Spike raster plot from an $N = 500$ cell sparse network with 6 connections per cell. The 379 cells which fire at least one spike during the period shown are plotted. The cells are ordered by k-means with 30 clusters. (d) Cross-correlation matrix corresponding to (c) with same conventions as (b).

attractor switching and have been studied in connection with deterministic intermittency[15]. Intermittency consists of laminar phases where the system orbits appear to be relatively regular, and bursts phases where the motion is quite violent and irregular. Interestingly a power-law distribution of state sojourn times was also observed in the hippocampal study of by Sasaki et al.[2] described above. Plenz and Thiagarajan[16] discuss cortical cell assemblies in the framework of scale free avalanches which are associated with intermittency[17].

The broad power-law distribution produces the temporal aspect of the complex identity-temporal patterns observed in the time series in Fig.2(c), however the fact that the cells show strong cross-correlation produces the spatial structure aspect. In the above we have shown how this structure can be revealed using the k-means sorting algorithm. By combining the spikes of cells in a cluster into a "cluster spike train" preserving each spikes' timing we can study the ISIs of cluster spike time series. However the k-means algorithm produces a different clustering depending on the initial choice of centroids. To control for this we perform the clustering many, here 200, times and combine the ISI time series so generated into a single distribution. The black circles in Fig.3(a) show the cluster ISI distribution after cells have been associated to clusters with the k-means algorithm with 10 clusters. The cluster ISI distribution, like the individual cell ISI distribution, also shows a power-law over several orders of magnitude. This implies clusters also burst in a multiple scale way. The slope of the power law is greater than the individual cell result and the cut-off is lower as would be expected when spike trains are combined. Nevertheless the distribution is still very broad. To demonstrate this we perform a bootstrap type test where rather than making each cluster spike train

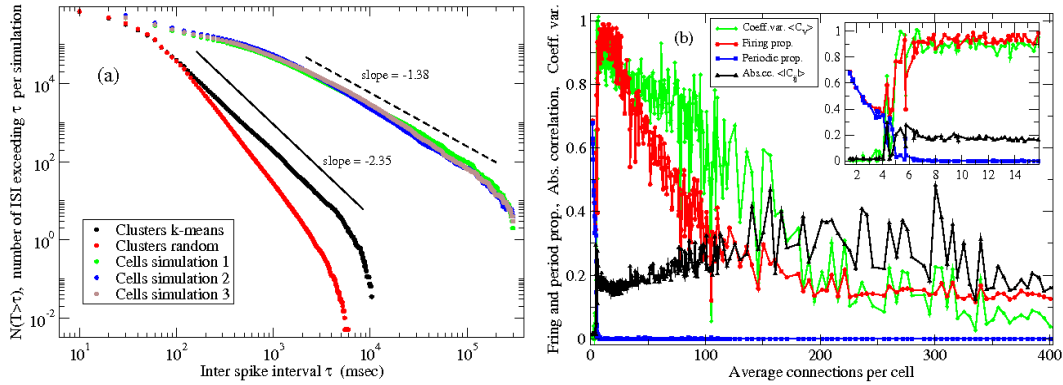

Figure 3: (a) Green, brown, blue: Three cell cumulative ISI distributions from 500 cell network simulations with 30 connections per cell, all cells combined. Log-log scale. The slope of the dashed line is $-1.38$. Black: ISI distribution for clusters formed by k-means algorithm corresponding to green single cell distribution. The slope of the solid line is $-2.35$. Red: ISI distribution for clusters formed from cells randomly corresponding to green single cell distribution. (b) Variation of connectivity for 500 cell networks. Inset shows low connectivity detail. Each point calculated from a different network simulation for observation period $t = 2000$ to $t = 12000$ msec. Red: Proportion of cells which fire at least one spike during the period. Blue: Proportion of cells firing periodically. Black: Average absolute cross-correlation $\langle |C_{ij}| \rangle$ between all cells in network calculated from rate time series constructed from counting spikes in moving window of size 2000 msec. Green: Coefficient of variation $\langle C_V \rangle$ of ISI distribution averaged across all cells in network rescaled by $1/3$.

from the cells associated to the cluster we perform the same k-means clustering to obtain correct cluster sizes but then scramble the cell indices, associating the cells to the clusters randomly. Again we do this 200 times and combine all the results into one cluster ISI distribution. The red circles in Fig.3(a) show this random cluster ISI distribution. The distribution is much narrower than the distribution obtained from the non-randomized k-means clustering. This demonstrates further that the time series have a clustered structure which can be revealed by the k-means algorithm and that the clusters produced have a larger periods of quiescence between bursting than would be expected from randomly associating cells, even when the cells themselves have power-law distributed ISIs. This broadened distribution produced by the clustering reflects the complex identity-temporal structure of the ordered spike time series figures such as shown in Fig.2(c).

The model has several parameters, in particular the connection probability $p$. How does the formation of switching assembly dynamics depend on the network connectivity? To study this we perform many numerical simulations while varying $p$. As described above the synaptic efficacy is rescaled by the connection probability so the total inhibition on each cell is fixed and therefore effects arise purely from variations in connectivity.

Fig.3(b) (red) shows the proportion of cells which fire at least one spike versus average connections per cell for 500 cell network simulations. This quantity shows a transition around 5 connections per cell to state where almost all the network is burst firing and then decays off to a plateau region at higher connectivity. Fig.3(b) (blue) shows the proportion of cells firing periodically. This is zero above the transition. Below the transition a large proportion of cells are not inhibited and firing periodically due to the excitatory cortical drive, while another large proportion are not firing at all, inhibited by the periodically firing group. At high connectivities however most cells receive similar inhibition levels which leaves a certain proportion firing. Fig.3(b) (green) shows the coefficient of variation $C_V$ of the single cell ISI distribution averaged across all cells and rescaled by $1/3$. $C_V$ is defined to be the ISI standard deviation normalized by the mean ISI. It is unity for Poisson processes. Below the transition $C_V$ is very low due to many periodic firing cells. At high connectivities it is also low and inspection of spike time series shows all cells firing with fairly regular ISIs. In intermediate regions however this quantity can become very large reflecting long periods of quiescent interrupted by high frequency bursting, as also reflected in the single cell ISI distributions in Fig.3(a). Fig.3(b) (black) shows the average absolute cross-correlation $\langle |C_{ij}| \rangle$ where $C_{ij}$ is the cross-correlation co-

efficient between cells $i$ and $j$ firing rate time series' and its absolute value is averaged across all cells. This quantity also shows the low connectivity transition but peaks around 200 connections per cell, where many cells are substantially cross-correlated (both positively and negatively). This is in accordance with the study of Wang and Buzsaki[5]. Fig.3(b) therefore displays an interesting regime between about 50 and 200 connections per cell where many cells are burst firing with long periods of quiescence but have substantial cross-correlation. It is in this regime that spike time series often show the complex identity-temporal patterns and switching cell assemblies exemplified in Fig.2(c).

## 4  Discussion

We have shown that inhibitory networks of biologically realistic spiking neurons obeying deterministic dynamical equations with sparse to intermediate connectivity can show bursting dynamics, complex identity-temporal patterns and form cell assemblies. The cells should be near a bifurcation point where even weak inhibition can cause them to become quiescent. The synapses should have a slower timescale, $\tau_g > 10$ in Eq.4, which produces a low pass filter of presynaptic spiking. This slow change in inhibition allows the bursting assembly dynamics since presynaptic cells do not instantly inhibit postsynaptic cells, but inhibition builds up gradually, allowing the formation of assemblies which eventually becoming strong enough to quench the postsynaptic cell activity.

At low connectivities sets of cells with sufficiently few and/or sufficiently weak connections between them will exist and these cells will fire together as an assembly due to the cortical excitation, if the rest of the network which inhibits them is sufficiently quiescent for a period. Such a set of weakly connected cells can be inhibited by another such set of weakly connected cells if each member of the first set is inhibited by a sufficient number of cells of the second set. When the second set ceases firing the first set will start to fire. These assemblies can exist in asymmetric closed loops which slowly switch active set. Multiple "frustrated" interlocking loops can exist where the slow switching of one loop will interfere with the dynamical switching of another loop; only when inhibition on one member set is removed will the loop be able to continue slow switching, producing a type of neural computation. Furthermore any given cell can be a member of several such sets of weakly connected cells, as also described by Assisi and Bazhenov[18]. This can explain the findings of Carrillo-Reid et.al.[1] who show some cells firing with only one assembly and other cells firing in multiple assemblies. These cross-coupled switching assemblies with partially shared members produce complex multiple timescale dynamics and identity-temporal patterning for appropriate connectivities.

Switching assemblies are most likely to be observed in networks of sparse to intermediate connectivities. This is consistent with WLC based attractor switching. Indeed networks with non-symmetric inhibitory connections which form closed circuits display WLC dynamics[3] and these will be likely to occur in networks with sparse to intermediate connectivities. The spike time series in Figs.2(a) and (c), indicate that cell assemblies switch non-randomly in sequence due to the deterministic attractor switching. This is in good agreement with Carrillo-Reid et al.[1] study of striatal dynamics and also with the Sasaki et al.[2] study of CA3 cell assemblies. Our time series and the cross-correlation matrices demonstrate that while most cells fire with only one particular assembly, some cells are shared between assemblies, as observed by Carrillo-Reid et al.[1]. We have shown that cells form assemblies of positively correlated cells and assemblies are negatively correlated with each other, in accordance with the similarity matrix results shown in Sasaki et al.[2].

Very interestingly cell assemblies are predominantly found in a connectivity regime appropriate for the striatum[6], where each cell is likely to be connected to about 100 cortically excited cells, suggesting the striatum may have adapted to be in this regime. Studies of spontaneous firing in the striatum also show very variable firing patterns with long periods of quiescence[14], as shown in our simulations at this connectivity. Based on studies of random striatal connectivity[6] we have simulated a random network without real spatial dimension. In support of this assumption Carrillo-Reid et al.[1] find that correlated activity is spatially distributed, noting that neurons firing synchronously could be hundreds of microns apart intermingled with silent cells.

Although we leave this point for future work the dynamics can also be affected by the details of the spiking. Detailed inspection of the spike raster plot in Fig.1(b) confirms three cells firing with identical frequency. Since these cells are driven by different levels of cortical excitation, the synchronization can only result from an entrainment produced by the spiking. This is possible in cells with close firing rates since the effect an inhibitory spike has on a post-synaptic cell depends on

the post-synaptic membrane potential[13, 19]. In this way the spiking can affect cluster formation dynamics and may prolong the lifetime of visits to quasi-stable periodic states. The coupling of assembly dynamics and spiking may be relevant for coding in the insect olfactory lobe for example[9].

The striatum is the main input structure to the basal ganglia (BG). Correlated activity in cortico-basal ganglia circuits is important in the encoding of movement, associative learning, sequence learning and procedural memory. Aldridge and Berridge[21] demonstrate that the striatum implements action syntax in rats grooming behaviour. BG may contain central pattern generators (CPGs) that activate innate behavioral routines, procedural memories, and learned motor programs[20] and recurrent alternating bursting is characteristic of cell assemblies included in CPGs[20]. WLC has been applied to modeling CPGs[11]. Our modeling suggests that complex switching dynamics based in the sparse striatal inhibitory network may allow the generation of cell assemblies which interface sensory driven cortical patterns to dynamical sequence generation. Further work is underway to demonstrate how these dynamics may be utilized in behavioural tasks recruiting the striatum.

## References

[1] Carrillo-Reid L, Tecuapetla F, Tapia D, Hernandez-Cruz A, Galarraga E, Drucker-Colin R, Bargas J. J.Neurophys. 99, 1435 (2008).

[2] Sasaki T, Matsuki N, Ikegaya Y. J.Neurosci. 27(3), 517-528 (2007). Sasaki T, Kimura R, Tsukamoto M, Matsuki N, Ikegaya Y. J.Physiol. 574.1, 195-208 (2006).

[3] Rabinovich MI, Huerta R, Volkovskii A, Abarbanel HDI, Stopfer M, Laurent G. J.Physiol. 94, 465 (2000). Rabinovich M, Volkovskii A, Lecanda P, Huerta R, Abarbanel HDI, Laurent G PRL 87,06:U149-U151 (2001). Rabinovich MI, Huerta R, Varona P, Afraimovich V. Biol. Cybern. 95:519-536 (2006). Nowotny T, Rabinovich MI. PRL 98,128106 (2007).

[4] Golomb D and Rinzel J. PRE 48, 4810-4814 (1993). Golomb D and Hansel D. Neur.Comp. 12, 1095-1139 (2000). Tiesinga PHE and Jose VJ. J.Comp.Neuro. 9(1):49-65 (2000).

[5] Wang X-J and Buzsaki G. J.Neurosci. 16(20):6402-6413 (1996).

[6] Wickens JR, Arbuthnott G, Shindou T. Prog. Brain Res. 160, 316 (2007).

[7] Rall WJ. Neurophys. 30, 1138-1168 (1967).

[8] Laurent G. Science 286, 723-728 (1999). Laurent G and Davidowitz H. Science 265, 1872-1875 (1994). Wehr M and Laurent G. Nature 384, 162-166 (1996). Laurent G, Stopfer M, Friedrich RW, Rabinovich MI, Abarbanel HDI Annu. Rev. Neurosci. 24:263-297 (2001).

[9] Bazhenov M, Stopfer M, Rabinovich MI, Huerta R, Abarbanel HDI, Sejnowski TJ, Laurent G. Neuron, 30, 553-567 (2001). Nowotny T, Huerta R, Abarbanel HDI, Rabinovich MI. Biol. Cybern 93, 436-446 (2005). Huerta R, Nowotny T, Garcia-Sanchez M, Abarbanel HDI. Neur.Comp. 16, 1601-1640 (2004).

[10] Rabinovich MI, Huerta R,Afraimovich VS. PRL 97, 188103 (2006). Rabinovich MI, Huerta R, Varona P, Afraimovich VS. PLoS Comput Biol 4(5): e1000072. doi:10.1371 (2008).

[11] Selverston A, Rabinovich M, Abarbanel H, Elson R, Sznes A, Pinto R, Huerta R, Varona P. J.Physiol. 94:357-374 (2000). Varona P, Rabinovich MI, Selverston AI, Arshavsky YI. Chaos 12(3) 672, (2003).

[12] Kaneko K, Tsuda I. Chaos 13(3), 926-936 (2003). K. Kaneko. Physica D 41, 137 (1990). I. Tsuda. Neural Networks 5, 313 (1992). Tsuda I, Fujii H, Tadokoro S, Yasuoka T, Yamaguti Y. J.Int.Neurosci, 3(2), 159-182 (2004). Fujii H and Tsuda I. Neurocomp. 58:151-157 (2004).

[13] Izhikevich E.M. Dynamical Systems in Neuroscience: The Geometry of .... MIT press (2005).

[14] Wilson CJ and Groves PM. Brain Research 220:67-80 (1981).

[15] Gluckenheimer J, Holmes P. Non-linear Oscillations,Dynamical Systems and Bifurcations of Vector Fields, Springer, Berlin (1983). Ott E. Chaos in Dynamical Systems. Cambridge, U.K.: Cambridge Univ. Press (2002). Pomeau Y and Manneville P. Commun. Math. Phys., 74, 189-197 (1980). Pikovsky A, J. Phys. A, 16, L109-L112, (1984).

[16] Plenz D, Thiagarajan TC. Trends Neurosci 30: 101-110, (2007).

[17] Bak P, Tang C, Wiesenfeld K. PRA 38, 364-374 (1988). Bak P, Sneppen K. PRL 71, 4083-4086 (1993).

[18] Assisi CG and Bazhenov MV. SfN 2007 abstract.

[19] Ermentrout B, Rep. Prog. Phys. 61, 353-430 (1998). van-Vreeswijk C, Abbott LF, Ermentrout B. J. Comp. Neuro., 1, 313-321 (1994).

[20] Grillner S, Hellgren J, Menard A, Saitoh K, Wikstrom MA. Trends Neurosci. 28: 364-370, (2005). Takakusaki K, Oohinata-Sugimoto J, Saitoh K, Habaguchi T. Prog Brain Res. 143: 231-237, (2004).

[21] Aldridge JW and Berridge KC. J. Neurosci., 18(7):2777-2787 (1998).

